# Rao-Blackwellised Particle Filtering via Data Augmentation

**Christophe Andrieu**

Statistics Group
University of Bristol
University Walk
Bristol BS8 1TW, UK
*C.Andrieu@bristol.ac.uk*

**Nando de Freitas**

Computer Science
UC Berkeley
387 Soda Hall, Berkeley
CA 94720-1776, USA
*jfgf@cs.berkeley.edu*

**Arnaud Doucet**

EE Engineering
University of Melbourne
Parkville, Victoria 3052
Australia
*doucet@ee.mu.oz.au*

## Abstract

In this paper, we extend the Rao-Blackwellised particle filtering method to more complex hybrid models consisting of Gaussian latent variables and discrete observations. This is accomplished by augmenting the models with artificial variables that enable us to apply Rao-Blackwellisation. Other improvements include the design of an optimal importance proposal distribution and being able to swap the sampling an selection steps to handle outliers. We focus on sequential binary classifiers that consist of linear combinations of basis functions, whose coefficients evolve according to a Gaussian smoothness prior. Our results show significant improvements.

## 1   Introduction

Sequential Monte Carlo (SMC) particle methods go back to the first publically available paper in the modern field of Monte Carlo simulation (Metropolis and Ulam 1949); see (Doucet, de Freitas and Gordon 2001) for a comprehensive review. SMC is often referred to as particle filtering (PF) in the context of computing filtering distributions for statistical inference and learning. It is known that the performance of PF often deteriorates in high-dimensional state spaces. In the past, we have shown that if a model admits partial analytical tractability, it is possible to combine PF with exact algorithms (Kalman filters, HMM filters, junction tree algorithm) to obtain efficient high dimensional filters (Doucet, de Freitas, Murphy and Russell 2000, Doucet, Godsill and Andrieu 2000). In particular, we exploited a marginalisation technique known as Rao-Blackwellisation (RB).

Here, we attack a more complex model that does not admit immediate analytical tractability. This probabilistic model consists of Gaussian latent variables and binary observations. We show that by augmenting the model with artificial variables, it becomes possible to apply Rao-Blackwellisation and optimal sampling strategies. We focus on the problem of sequential binary classification (that is, when the data arrives one-at-a-time) using generic classifiers that consist of linear combinations of basis functions, whose coefficients evolve according to a Gaussian smoothness

prior (Kitagawa and Gersch 1996). We have previously addressed this problem in the context of sequential fault detection in marine diesel engines (Højen-Sørensen, de Freitas and Fog 2000). This application is of great importance as early detection of incipient faults can improve safety and efficiency, as well as, help to reduce down-time and plant maintenance in many industrial and transportation environments.

## 2 Model Specification and Estimation Objectives

Let us consider the following binary classification model. Given at time $t = 1, 2, \ldots$ an input $x_t$ we observe $z_t \in \{0, 1\}$ such that

$$\Pr \left( z_t = 1 | x_t, \beta_t \right) = \Phi \left( f \left( x_t, \beta_t \right) \right), \qquad (1)$$

where $\Phi \left( u \right) = \frac{1}{\sqrt{2\pi}} \int_{-\infty}^{u} \exp \left( -a^2/2 \right) da$ is the cumulative function of the standard normal distribution. This is the so-called probit link. By convention, researchers tend to adopt a logistic (sigmoidal) link function $\varphi \left( u \right) = \left( 1 + \exp \left( -u \right) \right)^{-1}$. However, from a Bayesian computational point of view, the probit link has many advantages and is equally valid. The unknown function is modeled as

$$f \left( x_t, \beta_t \right) = \sum_{k=1}^{K} \beta_{t,k} \Psi_k \left( x_t \right) = \Psi^{\mathrm{T}} \left( x_t \right) \beta_t,$$

where we have assumed that the basis functions $\Psi \left( x_t \right) \triangleq \left( \Psi_1 \left( x_t \right), \ldots, \Psi_K \left( x_t \right) \right)^{\mathrm{T}}$ do not depend on unknown parameters; see (Andrieu, de Freitas and Doucet 1999) for the more general case. $\beta_t \triangleq \left( \beta_{t,1}, \ldots, \beta_{t,K} \right)^{\mathrm{T}} \in \mathbb{R}^K$ is a set of unknown time-varying regression coefficients. To complete the model, we assume that they satisfy

$$\beta_t = A_t \beta_{t-1} + B_t v_t, \ \beta_0 \sim \mathcal{N} \left( m_0, P_0 \right) \qquad (2)$$

where $v_t \overset{i.i.d.}{\sim} \mathcal{N} \left( 0, I_{n_v} \right)$ and $A$ and $B$ control model correlations and smoothing (regularisation). Typically $K$ is rather large, say 10 or 100, and the bases $\Psi_k \left( \cdot \right)$ are multivariate splines, wavelets or radial basis functions (Holmes and Mallick 1998).

### 2.1 Augmented Statistical Model

We augment the probabilistic model artificially to obtain more efficient sampling algorithms, as will be detailed in the next section. In particular, we introduce the set of independent variables $y_t$, such that

$$y_t = f \left( x_t, \beta_t \right) + n_t, \qquad (3)$$

where $n_t \overset{i.i.d.}{\sim} \mathcal{N} \left( 0, 1 \right)$, and define $z_t = \begin{cases} 1 & \text{if } y_t > 0, \\ 0 & \text{otherwise.} \end{cases}$ It is then easy to check that one has $\Pr \left( z_t = 1 | x_t, \beta_t \right) = \Phi \left( f \left( x_t, \beta_t \right) \right)$.

This data augmentation strategy was first introduced in econometrics by economics Nobel laureate Daniel McFadden (McFadden 1989). In the MCMC context, it has been used to design efficient samplers (Albert and Chib 1993). Here, we will show how to take advantage of it in an SMC setting.

### 2.2 Estimation objectives

Given, at time $t$, the observations $o_{1:t} \triangleq \left( x_{1:t}, z_{1:t} \right)$, any Bayesian inference is based on the posterior distribution[1] $P \left( d\beta_{0:t} | o_{1:t} \right)$. We are, therefore, interested in estimating sequentially in time this distribution and some of its features, such as

$\mathbb{E}(f(x_t, \beta_t)|o_{1:t})$ or the marginal predictive distribution at time $t$ for new input data $x_{t+1}$, that is $\Pr(z_{t+1} = 1|o_{1:t}, x_{t+1})$. The posterior density satisfies a time recursion according to Bayes rule, but it does not admit an analytical expression and, consequently, we need to resort to numerical methods to approximate it.

## 3   Sequential Bayesian Estimation via Particle Filtering

A straightforward application of SMC methods to the model (1)-(2) would focus on sampling from the high-dimensional distribution $P(d\beta_{0:t}|o_{1:t})$ (Højen-Sørensen et al. 2000). A substantially more efficient strategy is to exploit the augmentation of the model to sample only from the low-dimensional distribution $P(dy_{1:t}|o_{1:t})$. The low-dimensional samples allow us then to compute the remaining estimates analytically, as shown in the following subsection.

### 3.1   Augmentation and Rao-Blackwellisation

Consider the extended model defined by equations (1)-(2)-(3). One has

$$p(\beta_{0:t}|o_{1:t}) = \int p(\beta_{0:t}|x_{1:t}, y_{1:t}) \, p(y_{1:t}|o_{1:t}) \, dy_{1:t}.$$

Thus if we have a Monte Carlo approximation of $P(dy_{1:t}|o_{1:t})$ of the form

$$\widehat{P}_N(dy_{1:t}|o_{1:t}) = \sum_{i=1}^{N} w_t^{(i)} \delta_{y_{1:t}^{(i)}}(dy_{1:t}),$$

then $p(\beta_{0:t}|o_{1:t})$ can be approximated via

$$\widehat{p}_N(\beta_{0:t}|o_{1:t}) = \sum_{i=1}^{N} w_t^{(i)} p\left(\beta_{0:t}|x_{1:t}, y_{1:t}^{(i)}\right),$$

that is a mixture of Gaussians. From this approximation, one can estimate $\mathbb{E}(\beta_t|x_{1:t}, y_{1:t})$ and $\mathbb{E}(\beta_{t-L}|x_{1:t}, y_{1:t})$. For example, an estimate of the predictive distribution is given by

$$\widehat{\Pr}_N(z_{t+1} = 1|o_{1:t}, x_{t+1}) = \int \Pr(z_{t+1} = 1|y_{t+1}) \, \widehat{P}_N(dy_{1:t+1}|o_{1:t}, x_{t+1}) \quad (4)$$

$$= \sum_{i=1}^{N} w_t^{(i)} \mathbb{I}_{(0,+\infty)}\left(y_{t+1}^{(i)}\right),$$

where $y_{t+1}^{(i)} \sim P\left(dy_{t+1}|x_{1:t+1}, y_{1:t}^{(i)}\right)$. This shows that we can restrict ourselves to the estimation of $p(y_{1:t}|o_{1:t})$ for inference purposes.

In the SMC framework, we must estimate the "target" density $p(y_{1:t}|o_{1:t})$ pointwise up to a normalizing constant. By standard factorisation, one has $p(y_{1:t}|o_{1:t}) \propto \prod_{k=1}^{t} \Pr(z_k|y_k) \, p(y_k|x_{1:k}, y_{1:k-1})$, where $p(y_1|y_{1:0}, x_{1:0}) \triangleq p(y_1|x_1)$. Since $\Pr(z_k|y_k)$ is known, we only need to estimate $p(y_k|x_{1:k}, y_{1:k-1})$ up to a normalizing constant. This predictive density can be computed using the Kalman filter. Given $(x_{1:k}, y_{1:k-1})$, the Kalman filter equations are the following. Set $\beta_{0|0} = m_0$

and $\Sigma_{0|0} = \Sigma_0$, then for $t = 1, ..., k-1$ compute

$$
\begin{aligned}
\beta_{t|t-1} &= A_t \beta_{t-1|t-1} \\
\Sigma_{t|t-1} &= A_t \Sigma_{t-1|t-1} A_t^{\mathsf{T}} + B_t B_t^{\mathsf{T}} \\
S_t &= \Psi^{\mathsf{T}}(x_t) \Sigma_{t|t-1} \Psi(x_t) + 1 \\
y_{t|t-1} &= \Psi^{\mathsf{T}}(x_t) \beta_{t|t-1} \\
\beta_{t|t} &= \beta_{t|t-1} + \Sigma_{t|t-1} \Psi(x_t) S_t^{-1}(y_t - y_{t|t-1}) \\
\Sigma_{t|t} &= \Sigma_{t|t-1} - \Sigma_{t|t-1} \Psi(x_t) S_t^{-1} \Psi^{\mathsf{T}}(x_t) \Sigma_{t|t-1},
\end{aligned}
\tag{5}
$$

where $\beta_{t|t-1} \triangleq \mathbb{E}(\beta_t | x_{1:t-1}, y_{1:t-1})$, $\beta_{t|t} \triangleq \mathbb{E}(\beta_t | x_{1:t}, y_{1:t})$, $y_{t|t-1} \triangleq \mathbb{E}(y_t | x_{1:t}, y_{1:t-1})$, $\Sigma_{t|t-1} \triangleq cov(\beta_t | x_{1:t-1}, y_{1:t-1})$, $\Sigma_{t|t} \triangleq cov(\beta_t | x_{1:t}, y_{1:t})$ and $S_t \triangleq cov(y_t | x_{1:t}, y_{1:t-1})$. One obtains

$$
p(y_k | x_{1:k}, y_{1:k-1}) = \mathcal{N}(y_k; y_{k|k-1}, S_k).
\tag{6}
$$

## 3.2 Sampling Algorithm

In this section, we briefly outline the PF algorithm for generating samples from $p(dy_{1:t} | o_{1:t})$. (For details, please refer to our extended technical report at `http://www.cs.berkeley.edu/~jfgf/publications.html`.) Assume that at time $t-1$ we have $N$ particles $\{y_{1:t-1}^{(i)}\}_{i=1}^N$ distributed according to $P(dy_{1:t-1} | o_{1:t-1})$ from which one can get the following empirical distribution approximation

$$
P_N(dy_{1:t-1} | o_{1:t-1}) = \frac{1}{N} \sum_{i=1}^N \delta_{y_{1:t-1}^{(i)}}(dy_{1:t-1}).
$$

Various SMC methods can be used to obtain $N$ new paths $\{y_{1:t}^{(i)}\}_{i=1}^N$ distributed approximately according to $P(dy_{1:t} | o_{1:t})$. The most successful of these methods typically combine importance sampling and a selection scheme. Their asymptotic convergence ($N \rightarrow \infty$) is satisfied under mild conditions (Crisan and Doucet 2000).

Since the selection step is standard (Doucet et al. 2001), we shall concentrate on describing the importance sampling step. To obtain samples from $P(dy_{1:t} | o_{1:t})$, we can sample from a proposal distribution $Q(dy_{1:t})$ and weight the samples appropriately. Typically, researchers use the transition prior as proposal distribution (Isard and Blake 1996). Here, we implement an *optimal* proposal distribution, that is one that minimizes the variance of the importance weights $w(y_{1:t})$ *conditional upon* not modifying the path $y_{1:t-1}$. In our case, we have

$$
p(y_t | x_{1:t}, y_{1:t-1}, z_t) \propto
\begin{cases}
p(y_t | x_{1:t}, y_{1:t-1}) \, \mathbb{I}_{[0,+\infty)}(y_t) & \text{if } z_t = 1 \\
p(y_t | x_{1:t}, y_{1:t-1}) \, \mathbb{I}_{(-\infty,0)}(y_t) & \text{if } z_t = 0
\end{cases},
$$

which is a truncated Gaussian version of (6) of and consequently

$$
w(y_{1:t}) \propto \Pr(z_t | x_{1:t}, y_{1:t-1}) = \left(1 - \Phi\left(-\frac{y_{t|t-1}}{\sqrt{S_t}}\right)\right)^{z_t} \Phi\left(-\frac{y_{t|t-1}}{\sqrt{S_t}}\right)^{1-z_t}.
\tag{7}
$$

The algorithm is shown in Figure 1. (Please refer to our technical report for convergence details.)

**Remark 1** *When we adopt the optimal proposal distribution, the importance weight $w_t \propto \Pr(z_t | x_{1:t}, y_{1:t-1})$ does not depend on $y_t$. It is thus possible to carry out the selection step before the sampling step. The algorithm is then similar to the auxiliary variable particle filter of (Pitt and Shephard 1999). This modification to the original algorithm has important implications. It enables us to search for more*

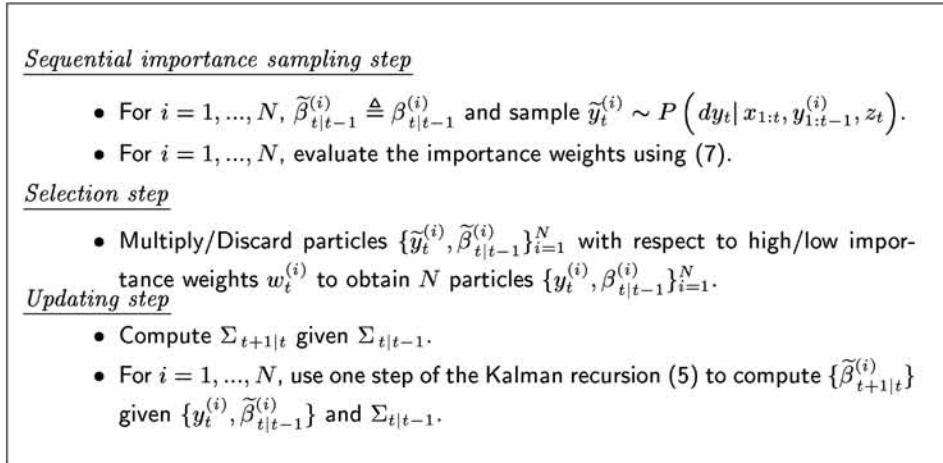

*Sequential importance sampling step*

- For $i = 1, ..., N$, $\widetilde{\beta}_{t|t-1}^{(i)} \triangleq \beta_{t|t-1}^{(i)}$ and sample $\widetilde{y}_t^{(i)} \sim P\left(dy_t \mid x_{1:t}, y_{1:t-1}^{(i)}, z_t\right)$.
- For $i = 1, ..., N$, evaluate the importance weights using (7).

*Selection step*

- Multiply/Discard particles $\{\widetilde{y}_t^{(i)}, \widetilde{\beta}_{t|t-1}^{(i)}\}_{i=1}^N$ with respect to high/low importance weights $w_t^{(i)}$ to obtain $N$ particles $\{y_t^{(i)}, \beta_{t|t-1}^{(i)}\}_{i=1}^N$.

*Updating step*

- Compute $\Sigma_{t+1|t}$ given $\Sigma_{t|t-1}$.
- For $i = 1, ..., N$, use one step of the Kalman recursion (5) to compute $\{\widetilde{\beta}_{t+1|t}^{(i)}\}$ given $\{y_t^{(i)}, \widetilde{\beta}_{t|t-1}^{(i)}\}$ and $\Sigma_{t|t-1}$.

Figure 1: RBPF for semiparametric binary classification.

*likely regions of the posterior at time $t-1$ using the information at time $t$ to generate better samples at time $t$. In practice, this increases the robustness of the algorithm to outliers and allows us to apply it in situations where the distributions are very peaked (e.g., econometrics and almost deterministic sensors and actuators).*

**Remark 2** *The covariance updates of the Kalman filter are outside the loop over particles. This results in substantial computational savings.*

## 4 Simulations

To compare our model, using the RBPF algorithm, to standard logistic and probit classification with PF, we generated data from clusters that change with time as shown in Figure 2. This data set captures the characteristics of a fault detection problem that we are currently studying. (For some results of applying PF to fault detection in marine diesel engines, please refer to (Højen-Sørensen et al. 2000). More results will become available once permission is granted.) This data cannot be easily separated with an algorithm based on a time-invariant model.

For the results presented here, we set the initial distributions to: $\beta_0 \sim \mathcal{N}(\mathbf{0}, 5\mathbf{I})$ and $y_0 \sim \mathcal{N}(\mathbf{0}, 5\mathbf{I})$. The process matrices were set to $A = I$ and $B = \delta I$, where $\delta^2 = 0.1$ is a smoothing parameter. The number of bases (cubic splines with random locations) was set to 10. (It is of course possible, when we have some data already, to initialise the bases locations so that they correspond to the input data. This trick for efficient classification in high dimensional input spaces is used in the support vector machines setting (Vapnik 1995).) The experiment was repeated with the number of particles varying between 10 and 400. Figure 3 shows the "value for money" summary plot. The new algorithm has a lower computational cost and shows a significant reduction in estimation variance. Note that the computation of the RBPF stays consistently low even for small numbers of particles. This has enabled us to apply the technique to large models consisting of hundreds of Bases using a suitable regulariser. Another advantage of PF algorithms for classification is that they yield entire probability estimates of class membership as shown in Figure 4.

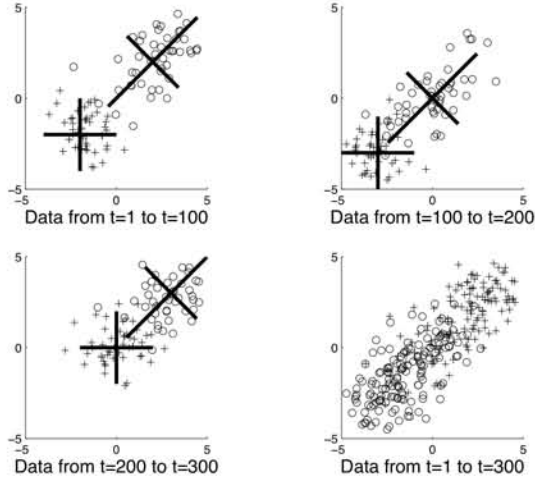

Figure 2: Time-varying data.

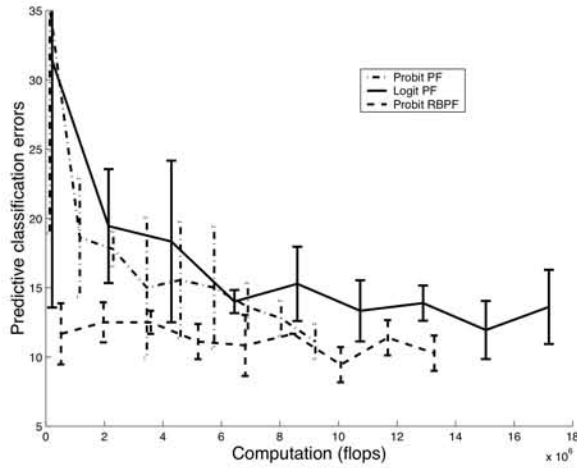

Figure 3: Number of classification errors as the number of particles varies between 10 and 400 (different computational costs). The algorithm with the augmentation trick (RBPF) is more efficient than standard PF algorithms.

## 5   Conclusions

In this paper, we proposed a dynamic Bayesian model for time-varying binary classification and an efficient particle filtering algorithm to perform the required computations. The efficiency of our algorithm is a result of data augmentation, Rao-Blackwellisation, adopting the optimal importance distribution, being able to swap the sampling and selection steps and only needing to update the Kalman filter means in the particles loop. This extends the realm of efficient particle filtering to the ubiquitous setting of Gaussian latent variables and binary observations. Extensions to n-ary observations, different link functions and estimation of the hyper-parameters can be carried out in the same framework.

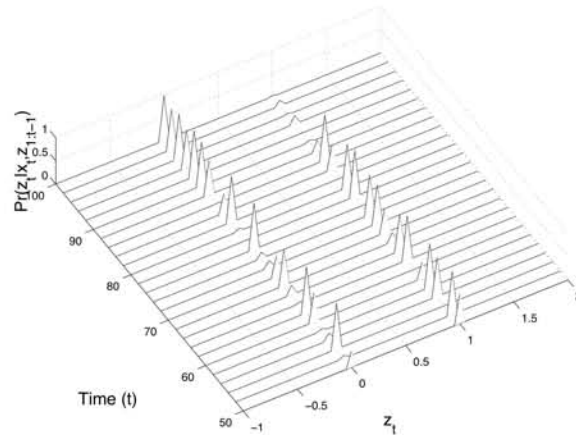

Figure 4: Predictive density.

## Footnotes

[1]For any $\theta$, we use $P \left( d\theta_{0:t} | o_{1:t} \right)$ to denote the distribution and $p \left( \theta_{0:t} | o_{1:t} \right)$ to denote the density, where $P \left( d\theta_{0:t} | o_{1:t} \right) = p \left( \theta_{0:t} | o_{1:t} \right) d\theta_{0:t}$. Also, $\theta_{0:t} \triangleq \{ \theta_0, \theta_1, \ldots, \theta_t \}$.

# References

Albert, J. and Chib, S. (1993). Bayesian analysis of binary and polychotomous response data, *Journal of the American Statistical Association* **88**(422): 669–679.

Andrieu, C., de Freitas, N. and Doucet, A. (1999). Sequential Bayesian estimation and model selection applied to neural networks, *Technical Report CUED/F-INFENG/TR 341*, Cambridge University Engineering Department.

Crisan, D. and Doucet, A. (2000). Convergence of sequential Monte Carlo methods, *Technical Report CUED/F-INFENG/TR 381*, Cambridge University Engineering Department.

Doucet, A., de Freitas, N. and Gordon, N. J. (eds) (2001). *Sequential Monte Carlo Methods in Practice*, Springer-Verlag.

Doucet, A., de Freitas, N., Murphy, K. and Russell, S. (2000). Rao blackwellised particle filtering for dynamic Bayesian networks, *in* C. Boutilier and M. Godszmidt (eds), *Uncertainty in Artificial Intelligence*, Morgan Kaufmann Publishers, pp. 176–183.

Doucet, A., Godsill, S. and Andrieu, C. (2000). On sequential Monte Carlo sampling methods for Bayesian filtering, *Statistics and Computing* **10**(3): 197–208.

Højen-Sørensen, P., de Freitas, N. and Fog, T. (2000). On-line probabilistic classification with particle filters, *IEEE Neural Networks for Signal Processing*, Sydney, Australia.

Holmes, C. C. and Mallick, B. K. (1998). Bayesian radial basis functions of variable dimension, *Neural Computation* **10**(5): 1217–1233.

Isard, M. and Blake, A. (1996). Contour tracking by stochastic propagation of conditional density, *European Conference on Computer Vision*, Cambridge, UK, pp. 343–356.

Kitagawa, G. and Gersch, W. (1996). *Smoothness Priors Analysis of Time Series*, Vol. 116 of *Lecture Notes In Statistics*, Springer-Verlag.

McFadden, D. (1989). A method of simulated momemts for estimation of discrete response models without numerical integration, *Econometrica* **57**: 995–1026.

Metropolis, N. and Ulam, S. (1949). The Monte Carlo method, *Journal of the American Statistical Association* **44**(247): 335–341.

Pitt, M. K. and Shephard, N. (1999). Filtering via simulation: Auxiliary particle filters, *Journal of the American Statistical Association* **94**(446): 590–599.

Vapnik, V. (1995). *The Nature of Statistical Learning Theory*, Springer-Verlag, New York.
